# Visual Cortex Circuitry and Orientation Tuning

**Trevor Mundel**
Department of Neurology
University of Chicago
Chicago, IL 60637
mundel@math.uchicago.edu

**Alexander Dimitrov**
Department of Mathematics
University of Chicago
Chicago, IL 60637
a-dimitrov@uchicago.edu

**Jack D. Cowan**
Departments of Mathematics and Neurology
University of Chicago
Chicago, IL 60637
cowan@math.uchicago.edu

## Abstract

A simple mathematical model for the large–scale circuitry of primary visual cortex is introduced. It is shown that a basic cortical architecture of recurrent local excitation and lateral inhibition can account quantitatively for such properties as orientation tuning. The model can also account for such local effects as cross–orientation suppression. It is also shown that non–local state–dependent coupling between similar orientation patches, when added to the model, can satisfactorily reproduce such effects as non–local iso–orientation suppression, and non–local cross–orientation enhancement. Following this an account is given of perceptual phenomena involving object segmentation, such as "pop-out", and the direct and indirect tilt illusions.

## 1 INTRODUCTION

The edge detection mechanism in the primate visual cortex (V1) involves at least two fairly well characterized circuits. There is a local circuit operating at sub–hypercolumn dimensions comprising strong orientation specific recurrent excitation and weakly orientation specific inhibition. The other circuit operates between hypercolumns, connecting cells with similar orientation preferences separated by several millimetres of cortical tissue. The horizontal connections which mediate this

circuit have been extensively studied. These connections are ideally structured to provide local cortical processes with information about the global nature of stimuli. Thus they have been invoked to explain a wide variety of context dependent visual processing. A good example of this is the tilt illusion (TI), where surround stimulation causes a misperception of the angle of tilt of a grating.

The interaction between such local and long–range circuits has also been investigated. Typically these experiments involve the separate stimulation of a cells receptive field (the *classical* receptive field or "center") and the immediate region outside the receptive field (the *non–classical* receptive field or "surround"). In the first part of this work we present a simple model of cortical center–surround interaction. Despite the simplicity of the model we are able to quantitatively reproduce many experimental findings. We then apply the model to the TI. We are able to reproduce the principle features of both the direct and indirect TI with the model.

## 2   PRINCIPLES OF CORTICAL OPERATION

Recent work with voltage–sensitive dyes (Blasdel, 1992) augments the early work of Hubel & Wiesel (1962) which indicated that clusters of cortical neurons corresponding to cortical columns have similar orientation preferences. An examination of local field potentials (Victor et al., 1994) which represent potentials averaged over cortical volumes containing many hundreds of cells show orientation preferences. These considerations suggest that the appropriate units for an analysis of orientation selectivity are the localized clusters of neurons preferring the same orientation. This choice of a population model immediately simplifies both analysis and computation with the model. For brevity we will refer to elements or edge detectors, however these are to be understood as referring to localized populations of neurons with a common orientation preference. We view the cortex as a lattice of hypercolumns, in which each hypercolumn comprises a continuum of iso–orientation patches distinguished by their preferred orientation $\phi$. All space coordinates refer to distances between hypercolumn centers. The population model we adopt throughout this work is a simplified form of the Wilson–Cowan equations.

### 2.1   LOCAL MODEL

Our local model is a ring ($\phi = -90^\circ to + 90^\circ$) of coupled iso–orientation patches and inhibitors with the following characteristics

- Weakly tuned orientation biased inputs to V1. These may arise either from slight orientation biases of lateral geniculate nucleus (LGN) neurons or from converging thalamocortical afferents

- Sharply tuned (space constant $\pm 7.5^\circ$) recurrent excitation between iso–orientation populations

- Broadly tuned inhibition to all iso–orientation populations with a cut-off of inhibition interactions at between $45^\circ$ and $60^\circ$ separation

The principle constraint is that of a critical balance between excitatory and inhibitory currents. Recent theoretical studies (Tsodyks & Sejnowski 1995; Vreeswijk & Sompolinsky 1996) have focused on this condition as an explanation for certain features of the dynamics of natural neuronal assemblies. These features include the irregular temporal firing patterns of cortical neurons, the sensitivity of neuronal assemblies in vivo to small fluctuations in total synaptic input and the distribution of firing rates in cortical networks which is markedly skewed towards low mean

rates. Vreeswijk & Sompolinsky demonstrate that such a balance emerges naturally in certain large networks of excitatory and inhibitory populations. We implement this critical balance by explicitly tuning the strength of connection weights between excitatory and inhibitory populations so that the system state is subcritical to a bifurcation point with respect to the relative strength of excitation/inhibition.

## 2.2 HORIZONTAL CONNECTIONS

We distinguish three potential patterns of horizontal connectivity

- connections between edge detectors along an axis parallel to the detectors preferred orientation (visuotopic connection)

- connections along an axis orthogonal to the detectors preferred orientation, with or without visuotopic connections

- radially symmetric connection to all detectors of the same orientation in surrounding hypercolumns

Recent experimental work in the tree shrew (Fitzpatrick et al., 1996) and preliminary work in the macaque (Blasdel, personal communication) indicate that visuotopic connection is the predominant pattern of long–range connectivity. This connectivity pattern allows for the following reduction in dimension of the problem for certain experimental conditions.

Consider the following experiment. A particular hypercolumn designated the "center" is stimulated with a grating at orientation $\phi$ resulting in a response from the $\phi$–edge detector. The region outside the receptive area of this hypercolumn (in the "surround") is also stimulated with a grating at some uniform orientation $\phi'$ resulting in responses from $\phi'$–edge detectors at each hypercolumn in the surround. In order to study the interactions between center and surround, then to first order approximation only the center hypercolumn and interaction with the surround along the $\phi$ visuotopic axis (defined by the center) and the $\phi'$ visuotopic axis (once again defined by the center) need be considered. In fact, except when $\phi = \phi'$ the effect of the center on the surround will be negligible in view of the modulatory nature of the horizontal connections detailed above. Thus we can reduce the problem (a priori three dimensional – one angle and two space dimensions) to two dimensions (one angle and one space dimension) with respect to a fixed center. This reduction is the key to providing a simple analysis of complex neurophysiological and psychophysical data.

# 3 RESULTS

## 3.1 CENTER–SURROUND INTERACTIONS

Although, we have modeled the state–dependence of the horizontal connections, many of the center–surround experiments we wish to model have not taken this dependence explicitly into account. In general the surround has been found to be suppressive on the center, which accords with the fact that the center is usually stimulated with high contrast stimuli. A typical example of the surround suppressive effect is shown in figure 1.

The basic finding is that stimulation in the surround of a visual cortical cell's receptive field generally results in a suppression of the cell's tuning response that is maximal for surround stimulation at the orientation of the cell's peak tuning

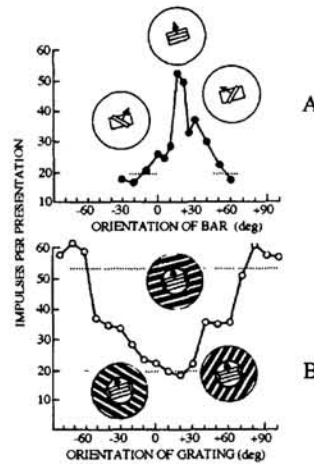

Figure 1: Non-local effects on orientation tuning - experimental data. Response to constant center stimulation at $15°$ and surround stimulation at angles $[-90°, 90°]$ (open circles), Local tuning curve (filled circles). Redrawn from Blakemore and Tobin (1972)

response and falls off with stimulation at other orientations in a characteristic manner. Further examples of surround suppression can be found in the paper of Sillito et al. (1995). Figure 2 depicts simulations in which long–range connections to local inhibitory populations are strong compared to connections to local excitatory populations.

These experiments and simulations appear to conflict with the consistent experimental finding that stimulating a hypercolumn with an orthogonal stimulus suppresses the response to the original stimulus.

The relevant results can be summarised as follows: cross–orientation suppression (with orthogonal gratings) originates within the receptive field of most cells examined and is a consistent finding in both complex and simple cells. The degree of suppression depends linearly on the size of the orthogonal grating up to a critical dimension which is smaller than the classical receptive field dimension. It is possible to suppress a response to the baseline firing rate by either increasing the size or the contrast of the orthogonal grating.

The model outlined earlier can account for all these observations, and similar measurements recently described by Sillitto et. al. (1995), in a strikingly simple fashion in the setting of single mode bifurcations. Orthogonal inputs are of the form $a/2[1 + \cos 2s(\phi - \phi_0)] + b/2[1 + \cos 2s(\phi - \phi_0 + 90°)]$, where $a$ and $b$ are amplitudes with $a > b$ and $\phi \in [-90°, 90°]$. By simple trigonometry this simplifies to $(a+b)/2 + (a-b)/2 \cos 2s(\phi - \phi_0)$ Thus the input of amplitude $b$ reduces the amplitude of the orthogonal input and hence gives rise to a smaller response. This is then the mechanism by which local double orthogonal stimulation leads to suppression.

The center–surround case is different in that the orthogonal input originates from the horizontal connections and (in the suppressive setting) is input primarily to the orthogonal inhibitory population. It can be shown rigorously that for small amplitude stimuli this is equivalent to an orthogonal input to the excitatory population with opposite sign. Thus we have a total input $(a + b)/2[1 + \cos(2s(\phi - \phi_0)]$ where $b$ arises from the horizontal input and hence increases the amplitude of the fundamental component of the input.

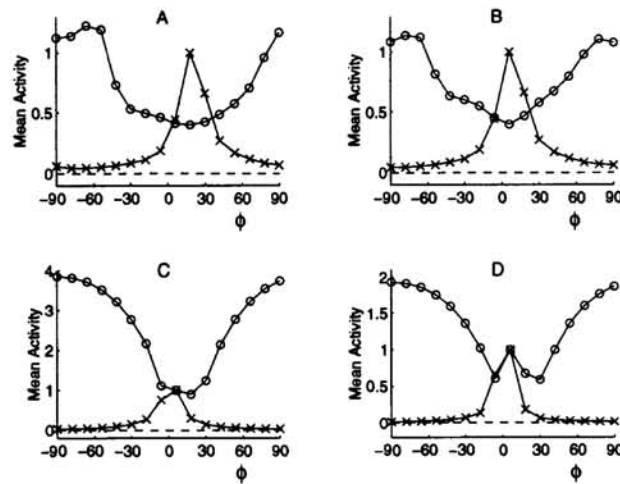

Figure 2: Non-local effects on orientation tuning . (o-o-o) = Center response to preferred orientation at different surround orientations, (x-x-x) = Center orientation tuning without surround stimulation, (—) = Center response to surround stimulation alone. A - response of population with 20° orientation preference. B, C and D - response of populations with 5° orientation preference.

More realistically, multiple modes may bifurcate simultaneously, though in the small amplitude linear regime where orientation preference is determined these can all be treated separately in the manner detailed above and lead to the same result.

## 3.2 POPOUT AND GAIN CONTROL

"Stimulus" dependent horizontal connections have recently been used to model a possible physiological correlate for the psychophysical phenomena of " pop–out " and enhancement (Stemmler, Usher & Niebur, 1995). In this model the horizontal input to excitatory neurons is via sodium channels which are dependent in the conventional manner on the differential between the membrane voltage and the sodium equilibrium potential. For such sodium channels the synaptic currents are attenuated as the membrane depolarizes towards the sodium equilibrium potential. This effect is opposite to that observed for the sodium channels mediating the horizontal input (Hirsch & Gilbert, 1991) which show increased synaptic currents as the membrane is depolarized. Thus although this model reproduces the phenomena described above it does not do so on the basis of the known physiology. We have confirmed with our model that weak stimulus enhancement and strong stimulus pop–out can be modelled with a variety of formulations for the excitatory neuron horizontal input, including a formulation which attenuates with increasing activity as in the model of Stemmler et. al.

It is interesting to note that one overall effect of the horizontal connections is to act as a type of gain control uniformizing the local response over a range of stimulus strengths. This gain control function has been discussed by Somers, Nelson & Sur (1994).

## 3.3 THE TILT ILLUSION

The tilt illusion (TI) is one of the basic orientation–based visual illusion. A TI occurs when viewing a test line against an inducing grating of uniformly oriented lines with an angle of $\theta$ between the orientation of the test line and the inducing

grating. Two components of the TI have been described (Wenderoth & Johnstone, 1988), the *direct* TI where the test line appears to be repelled by the grating–the orientation differential appears increased, and the *indirect* TI where the test line appears attracted to the orientation of the grating–the orientation differential appears decreased. Figure 3 depicts a typical plot of magnitude of the tilt effect versus the angle differential between inducing grating and test line reproduced from Wenderoth & Beh (1977). The TI thus provides compelling evidence that local

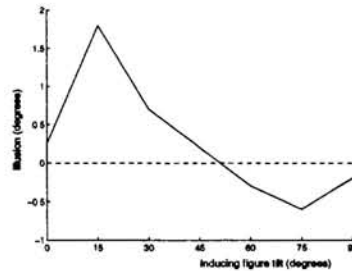

Figure 3: Direct (positive) and indirect (negative) tilt effects

detection of edges is dependent on information from more distant points in the visual field. It is generally believed, that the direct TI is due to lateral inhibition between cortical neurones (Wenderoth & Johnstone, 1988: Carpenter & Blakemore, 1973). It has been postulated that the indirect TI occurs at a higher level of visual processing. We show here that both the direct and indirect TI are a consequence of the lateral and local connections in our model.

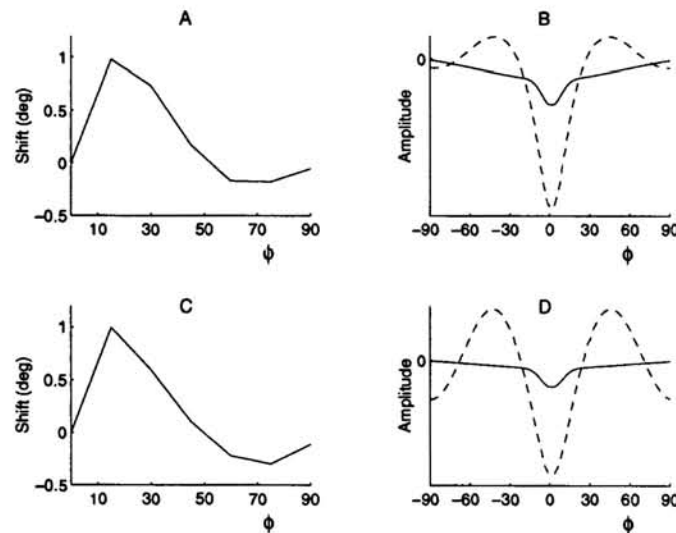

Figure 4: Model simulations of the tilt effect (A and C). B and D show the corresponding kernels mediating long–range interactions. Solid lines indicate the absolute kernel and dashed lines indicate the effective kernel

In figure 4 we give examples of the TI obtained from the model system. The effective kernels for long–range interactions are obtained by filtering the absolute kernels with the local filter which has a band–pass characteristic. It is this effective kernel which determines the tilt effect in keeping with our simulations and analysis which show that orientation preference is determined at the small amplitude linear stage of system development.

# 4 SUMMARY AND DISCUSSION

We have shown that a very simple center–surround organization, operating in the orientation domain can successfully account for a wide range of neurophysiological and psychophysical phenomena, all involving the effects of visual context on the responses of assemblies of spiking neurons. We expect to be able to show that such an organization can be seen in many parts of the cortex, and that it plays an important role in many forms of information processing in the brain.

## Acknowledgements

Supported in part by Grant # 96–24 from the James S. McDonnell Foundation.

## References

C. Blakemore and E.A. Tobin, Lateral Inhibition Between Orientation Detectors in the Cat's Visual Cortex, Exp. Brain Res., **15**, 439–440, (1972)

G.G. Blasdel, Orientation selectivity, preference, and continuity in monkey striate cortex, J. Neurosci. **12** No 8, 3139–3161 (1992)

R.H.S. Carpenter and C. Blakemore, Interactions between orientations in human vision, Expl. Brain. Res., **18**, 287–303, (1973)

G.C. DeAngelis, J.G. Robson, I. Ohzawa and R.D. Freeman, Organization of Suppression in Receptive Fields of Neurons in Cat Visual Cortex, J. Neurophysiol., **68** No 1, 144–163, (1992)

D. Fitzpatrick, The Functional Organization of Local Circuits in Visual Cortex: Insights from the Study of Tree Shrew Striate Cortex, Cerebral Cortex **6**, 329–341, (1996)

J.D. Hirsch and C.D. Gilbert, Synaptic physiology of horizontal connections in the cat's visual cortex, J. Neurosci., **11**, 1800–1809, (1991)

A.M. Sillito, K.L. Grieve, H.E. Jones, J. Cudeiro & J. Davis, Visual cortical mechanisms detecting focal orientation discontinuities, Nature, 378, 492–496, (1995)

D.C. Somers, S. Nelson and M. Sur, Effects of long–range connections on gain control in an emergent model of visual cortical orientation selectivity, Soc. Neurosci.,**20**, 646.7, (1994)

M. Stemmler, M. Usher and E. Niebur, Lateral Interactions in Primary Visual Cortex: A Model Bridging Physiology and Psychophysics, Science, **269**, 1877–1880, (1995)

M.V. Tsodyks and T. Sejnowski, Rapid state switching in balanced cortical network models, Network, **6** No 2, 111–124, (1995)

J.D. Victor, K. Purpura, E. Katz and B. Mao, Population encoding of spatial frequency, orientation and color in macaque V1, J. Neurophysiol., **72** No 5, (1994)

C. Vreeswijk and H. Sompolinsky, Chaos in neuronal networks with balanced excitatory and inhibitory activity. Science 274, 1724–1726, (1996)

P. Wenderoth and H. Beh, Component analysis of orientation illusions, Perception, **6** 57–75, (1977)

P. Wenderoth and S. Johnstone, The different mechanisms of the direct and indirect tilt illusions, Vision Res., **28** No 2, 301–312, (1988)